# The Conjoint Effect of Divisive Normalization and Orientation Selectivity on Redundancy Reduction in Natural Images

**Fabian Sinz**
MPI for Biological Cybernetics
72076 Tübingen, Germany
fabee@tuebingen.mpg.de

**Matthias Bethge**
MPI for Biological Cybernetics
72076 Tübingen, Germany
mbethge@tuebingen.mpg.de

## Abstract

Bandpass filtering, orientation selectivity, and contrast gain control are prominent features of sensory coding at the level of V1 simple cells. While the effect of bandpass filtering and orientation selectivity can be assessed within a linear model, contrast gain control is an inherently nonlinear computation. Here we employ the class of $L_p$ elliptically contoured distributions to investigate the extent to which the two features—orientation selectivity and contrast gain control—are suited to model the statistics of natural images. Within this framework we find that contrast gain control can play a significant role for the removal of redundancies in natural images. Orientation selectivity, in contrast, has only a very limited potential for redundancy reduction.

## 1 Introduction

It is a long standing hypothesis that sensory systems are adapted to the statistics of their inputs. These natural signals are by no means random, but exhibit plenty of regularities. Motivated by information theoretic principles, Attneave and Barlow suggested that one important purpose of this adaptation in sensory coding is to model and reduce the redundancies [4; 3] by transforming the signal into a statistically independent representation.

The problem of redundancy reduction can be split into two parts: (i) finding a good statistical model of the natural signals and (ii) a way to map them into a factorial representation. The first part is relevant not only to the study of biological systems, but also to technical applications such as compression and denoising. The second part offers a way to link neural response properties to computational principles, since neural representations of natural signals must be advantageous in terms of redundancy reduction if the hypothesis were true. Both aspects have been extensively studied for natural images [2; 5; 8; 19; 20; 21; 24]. In particular, it has been shown that applying Independent Component Analysis (ICA) to natural images consistently and robustly yields filters that are localized, oriented and show bandpass characteristics [19; 5]. Since those features are also ascribed to the receptive fields of neurons in the primary visual cortex (V1), it has been suggested that the receptive fields of V1 neurons are shaped to form a minimally redundant representation of natural images [5; 19].

From a redundancy reduction point of view, ICA offers a small but significant advantage over other linear representations [6]. In terms of density estimation, however, it is a poor model for natural images since already a simple non-factorial spherically symmetric model yields a much better fit to the data [10].

Recently, Lyu and Simoncelli proposed a method that converts any spherically symmetric distribution into a (factorial) Gaussian (or Normal distribution) by using a non-linear transformation of the

norm of the image patches [17]. This yields a non-linear redundancy reduction mechanism, which exploits the superiority of the spherically symmetric model over ICA. Interestingly, the non-linearity of this Radial Gaussianization method closely resembles another feature of the early visual system, known as contrast gain control [13] or divisive normalization [20]. However, since spherically symmetric models are invariant under orthogonal transformations, they are agnostic to the particular choice of basis in the whitened space. Thus, there is no role for the shape of the filters in this model.

Combining the observations from the two models of natural images, we can draw two conclusions: On the one hand, ICA is not a good model for natural images, because a simple spherically symmetric model yields a much better fit [10]. On the other hand, the spherically symmetric model in Radial Gaussianization cannot capture that ICA filters do yield a higher redundancy reduction than other linear transformations. This leaves us with the questions whether we can understand the emergence of oriented filters in a more general redundancy reduction framework, which also includes a mechanism for contrast gain control.

In this work we address this question by using the more general class of $L_p$-spherically symmetric models [23; 12; 15]. These models are quite similar to spherically symmetric models, but do depend on the particular shape of the linear filters. Just like spherically symmetric models can be non-linearly transformed into isotropic Gaussians, $L_p$-spherically symmetric models can be mapped into a unique class of factorial distributions, called $p$-generalized Normal distributions [11]. Thus, we are able to quantify the influence of orientation selective filters and contrast gain control on the redundancy reduction of natural images in a joint model.

## 2    Models and Methods

### 2.1    Decorrelation and Filters

All probabilistic models in this paper are defined on whitened natural images. Let $C$ be the covariance matrix of the pixel intensities for an ensemble $x_1, ..., x_m$ of image patches, then $C^{-\frac{1}{2}}$ constitutes the symmetric whitening transform. Note that all vectors $y = VC^{-\frac{1}{2}}x$, with $V$ being an orthogonal matrix, have unit covariance. $VC^{-\frac{1}{2}}$ yield the linear filters that are applied to the raw image patches before feeding them in the probabilistic models described below. Since any decorrelation transform can be written as $VC^{-\frac{1}{2}}$, the choice of $V$ determines the shape of the linear filters. In our experiments, we use three different kinds of $V$:

**SYM** The simplest choice is $V_{\text{SYM}} = I$, i.e. $y = C^{-\frac{1}{2}}x$ contains the coefficients in the symmetric whitening basis. From a biological perspective, this case is interesting as the filters resemble receptive fields of retinal ganglion cells with center-surround properties.

**ICA** The filters $V_{\text{ICA}}$ of ICA are determined by maximizing the non-Gaussanity of the marginal distributions. For natural image patches, ICA is known to yield orientation selective filters in resemblance to V1 simple cells. While other orientation selective bases are possible, the filters defined by $V_{\text{ICA}}$ correspond to the optimal choice for redundancy reduction under the restriction to linear models.

**HAD** The coefficients in the basis $V_{\text{HAD}} = \frac{1}{\sqrt{m}}HV_{\text{ICA}}$, with $H$ denoting an arbitrary Hadamard matrix, correspond to a sum over the different ICA coefficients, each possibly having a flipped sign. Hadamard matrices are defined by the two properties $H_{ij} = \pm 1$ and $HH^\top = mI$. This case can be seen as the opposite extreme to the case of ICA. Instead of running an independent search for the most Gaussian marginals, the central limit theorem is used to produce the most Gaussian components by using the Hadamard transformation to mix all ICA coefficients with equal weight resorting to the independence assumption underlying ICA.

### 2.2    $L_p$-spherically Symmetric Distributions

The contour lines of spherically symmetric distributions have constant Euclidean norm. Similarly, the contour lines of $L_p$-spherically symmetric distributions have constant $p$-norm[1] $||y||_p :=$

$\sqrt[p]{\sum_{i=1}^{n} |y_i|^p}$ The set of vectors with constant $p$-norm $\mathbb{S}_p^{n-1}(r) := \{\boldsymbol{y} \in \mathbb{R}^n : ||\boldsymbol{y}||_p = r, \ p > 0, \ r > 0\}$ is called $p$-sphere of radius $r$. Different examples of $p$-spheres are shown along the coordinate axis of Figure 1. For $p \neq 2$ the distribution is not invariant under arbitrary orthogonal transformations, which means that the choice of the basis $\boldsymbol{V}$ can make a difference in the likelihood of the data.

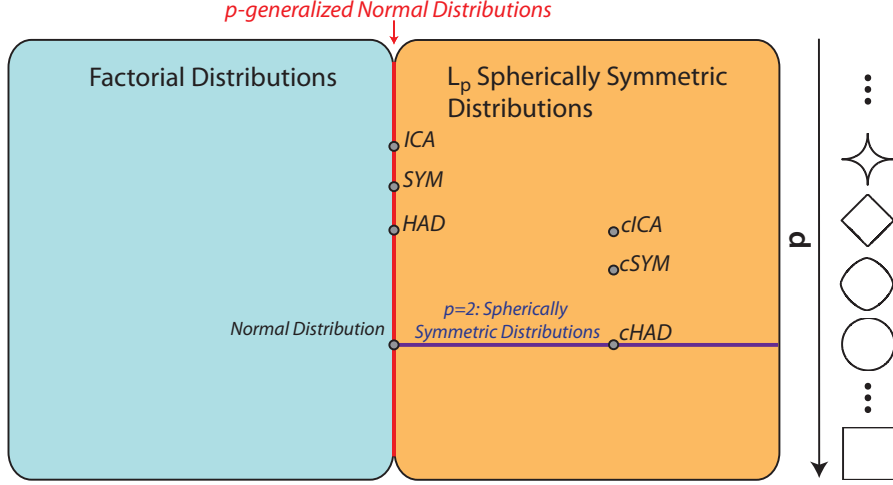

Figure 1: The spherically symmetric distributions are a subset of the $L_p$-spherical symmetric distributions. The right shapes indicate the iso-density lines for the different distributions. The Gaussian is the only $L_2$-spherically symmetric distribution with independent marginals. Like the Gaussian distribution, all $p$-generalized Normal distributions have independent marginals. *ICA, SYM, ...* denote the models used in the experiments below.

A multivariate random variable $Y$ is called $L_p$-spherically symmetric distributed if it can be written as a product $Y = RU$, where $U$ is uniformly distributed on $\mathbb{S}_p^{n-1}(1)$ and $R$ is a univariate non-negative random variable with an arbitrary distribution [23; 12]. Intuitively, $R$ corresponds to the radial component, i. e. the length $||\boldsymbol{y}||_p$ measured with the $p$-norm. $U$ describes the directional components in a polar-like coordinate system (see Extra Material). It can be shown that this definition is equivalent to the density $\varrho(\boldsymbol{y})$ of $Y$ having the form $\varrho(\boldsymbol{y}) = f(||\boldsymbol{y}||_p^p)$ [12]. This immediately suggests two ways of constructing an $L_p$-spherically symmetric distribution. Most obviously, one can specify a density $\varrho(\boldsymbol{y})$ that has the form $\varrho(\boldsymbol{y}) = f(||\boldsymbol{y}||_p^p)$. An example is the $p$-generalized Normal distribution (gN) [11]

$$\varrho(\boldsymbol{y}) \quad = \quad \frac{p^n}{\Gamma^n\left(\frac{1}{p}\right)(2\sigma^2)^{\frac{n}{p}}2^n} \exp\left(-\frac{\sum_{i=1}^{n} |y_i|^p}{2\sigma^2}\right) = f(||\boldsymbol{y}||_p^p). \tag{1}$$

Analogous to the Gaussian being the only factorial spherically symmetric distribution [1], this distribution is the only $L_p$-spherically symmetric distribution with independent marginals [22]. For the $p$-generalized Normal, the marginals are members of the exponential power family.

In our experiments, we will use the $p$-generalized Normal to model linear marginal independence by fitting it to the coefficients of the various bases in whitened space. Since this distribution is sensitive to the particular filter shapes for $p \neq 2$, we can assess how well the distribution of the linearly transformed image patches is matched by a factorial model.

An alternative way of constructing an $L_p$-spherically symmetric distribution is to specify the radial distribution $\varrho_r$. One example, which will be used later, is obtained by choosing a mixture of Log-Normal distributions (RMixLogN). In Cartesian coordinates, this yields the density

$$\varrho(\boldsymbol{y}) \quad = \quad \frac{p^{n-1}\Gamma\left(\frac{n}{p}\right)}{2^n\Gamma^n\left(\frac{1}{p}\right)} \sum_{k=1}^{K} \frac{\eta_k}{||\boldsymbol{y}||_p^n \sigma_k \sqrt{2\pi}} \exp\left(-\frac{(\log||\boldsymbol{y}||_p - \mu_k)^2}{2\sigma_k^2}\right). \tag{2}$$

An immediate consequence of any $L_p$-spherically symmetric distribution being specified by its radial density is the possibility to change between any two of those distributions by transforming the radial component with $(\mathcal{F}_2^{-1} \circ \mathcal{F}_1)(||\boldsymbol{y}||_p)$, where $\mathcal{F}_1$ and $\mathcal{F}_2$ are cumulative distribution functions (cdf) of the source and the target density, respectively. In particular, for a fixed $p$, any $L_p$-spherically symmetric distribution can be transformed into a factorial one by the transform

$$\boldsymbol{z} = g(\boldsymbol{y}) \cdot \boldsymbol{y} = \frac{(\mathcal{F}_2^{-1} \circ \mathcal{F}_1)(||\boldsymbol{y}||_p)}{||\boldsymbol{y}||_p}\boldsymbol{y}.$$

This transform closely resembles contrast gain control models for primary visual cortex [13; 20], which use a different gain function having the form $\tilde{g}(\boldsymbol{y}) = \frac{1}{c+r}$ with $r = ||\boldsymbol{y}||_2^2$ [17].

We will use the distribution of equation (2) to describe the joint model consisting of a linear filtering step followed by a contrast gain control mechanism. Once, the linear filter responses in whitened space are fitted with this distribution, we non-linearly transform it into a the factorial $p$-generalized Normal by the transformation $g(\boldsymbol{y}) \cdot \boldsymbol{y} = (\mathcal{F}_{\mathsf{gN}}^{-1} \circ \mathcal{F}_{\mathsf{RMixLogN}})(||\boldsymbol{y}||_p)/||\boldsymbol{y}||_p \cdot \boldsymbol{y}$.

Finally, note that because a $L_p$-spherically symmetric distribution is specified by its univariate radial distribution, fitting it to data boils down to estimating the univariate density for $R$, which can be done efficiently and robustly.

## 3 Experiments and Results

### 3.1 Dataset

We use the dataset from the Bristol Hyperspectral Images Database [7], which was already used in previous studies [25; 16]. All images had a resolution of $256 \times 256$ pixels and were converted to gray level by averaging over the channels. From each image circa 5000 patches of size $15 \times 15$ pixels were drawn at random locations for training (circa 40000 patches in total) as well as circa 6250 patches per image for testing (circa 50000 patches in total). In total, we sampled ten pairs of training and test sets in that way. All results below are averaged over those. Before computing the linear filters, the DC component was projected out with an orthogonal transformation using a QR decomposition. Afterwards, the data was rescaled in order to make whitening a volume conserving transformation (a transformation with determinant one) since those transformations leave the entropy unchanged.

### 3.2 Evaluation Measure

In all our experiments, we used the Average Log Loss (ALL) to assess the quality of the fit and the redundancy reduction achieved. The $\text{ALL} = \frac{1}{n}\mathbb{E}_\varrho[-\log_2 \hat{\varrho}(\boldsymbol{y})] \approx \frac{1}{mn}\sum_{k=1}^{m} -\log_2 \hat{\varrho}(\boldsymbol{y})$ is the negative mean log-likelihood of the model distribution under the true distribution. If the model distribution matches the true one, the ALL equals the entropy. Otherwise, the difference between the ALL and the entropy of the true distribution is exactly the Kullback-Leiber divergence between the two. The difference between the ALLs of two models equals the reduction in multi-information (see Extra Material) and can therefore be used to quantify the amount of redundancy reduction.

### 3.3 Experiments

We fitted the $L_p$-spherically symmetric distributions from equations (1) and (2) to the image patches in the bases HAD, SYM, and ICA by a maximum likelihood fit on the radial component. For the mixture of Log-Normal distributions, we used EM for a mixture of Gaussians on the logarithm of the $p$-norm of the image patches.

For each model, we computed the maximum likelihood estimate of the model parameters and determined the best value for $p$ according to the ALL in bits per component on a training set. The final ALL was computed on a separate test set.

For ICA, we performed a gradient descent over the orthogonal group on the log-likelihood of a product of independent exponential power distributions, where we used the result of the FastICA algorithm by Hyvärinen et al. as initial starting point [14]. All transforms were computed separately for each training set.

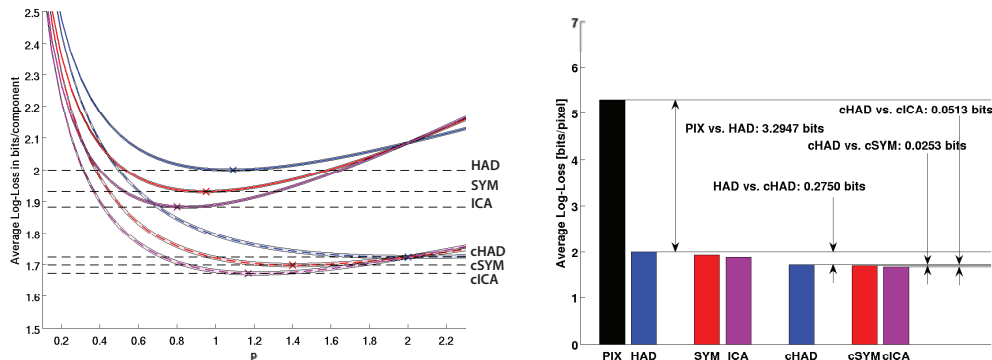

Figure 2: ALL in bits per component as a function of $p$. The linewidth corresponds to the standard deviation over ten pairs of training and test sets. *Left:* ALL for the bases HAD, SYM and ICA under the $p$-generalized Normal (HAD, SYM, ICA) and the factorial $L_p$-spherically symmetric model with the radial component modeled by a mixture of Log-Normal distributions (cHAD, cSYM, cICA). *Right:* Bar plot for the different ALL indicated by horizontal lines in the left plot.

In order to compare the redundancy reduction of the different transforms with respect to the pixel basis (PIX), we computed a non-parametric estimate of the marginal entropies of the patches before the DC component was projected out [6]. Since the estimation is not bound to a particular parametric model, we used the mean of the marginal entropies as an estimate of the average log-loss in the pixel representation.

## 3.4 Results

Figure 2 and Table 1 show the ALL for the bases HAD, SYM, and ICA as a function of $p$. The upper curve bundle represents the factorial $p$-generalized Normal model, the lower bundle the non-factorial model with the radial component modeled by a mixture of Log-Normal distributions with five mixtures. The ALL for the factorial models always exceeds the ALL for the non-factorial models. At $p = 2$, all curves intersect, because all models are invariant under a change of basis for that value. Note that the smaller ALL of the non-factorial model cannot be attributed to the mixture of Log-Normal distributions having more degrees of freedom. As mentioned in the introduction, the $p$-generalized Normal is the only factorial $L_p$-spherically symmetric distribution [22]. Therefore, marginal independence is such a rigid assumption that the output scale is the only degree of freedom left.

From the left plot in Figure 2, we can assess the influence of the different filter shapes and contrast gain control on the redundancy reduction of natural images. We used the best ALL of the HAD basis under the $p$-generalized Normal as a baseline for a whitening transformation without contrast gain control (HAD). Analogously, we used the best ALL of the HAD basis under the non-factorial model as a baseline for a pure contrast gain control model (cHAD). We compared these values to the best ALL obtained by using the SYM and the ICA basis under both models. Because the filters of SYM and ICA resemble receptive field properties of retinal ganglion cells and V1 simple cells, respectively, we can assess their possible influence on the redundancy reduction with and without contrast gain control. The factorial model corresponds to the case without contrast gain control (SYM and ICA). Since we have shown that the non-factorial model can be transformed into a factorial one by a $p$-norm based divisive normalization operation, these scores correspond to the cases with contrast gain control (cSYM and cICA). The different cases are depicted by the horizontal lines in Figure 2.

As already reported in other works, plain orientation selectivity adds only very little to the redundancy reduction achieved by decorrelation and is less effective than the baseline contrast gain control model [10; 6; 17]. If both orientation selectivity and contrast gain control are combined (cICA) it is possible to achieve about $9\%$ extra redundancy reduction in addition to baseline whitening

| | Absolute Difference [Bits/Comp.] | Relative Difference [% wrt. cICA] |
|---|---|---|
| HAD - PIX | $-3.2947 \pm 0.0018$ | $91.0016 \pm 0.0832$ |
| SYM- PIX | $-3.3638 \pm 0.0022$ | $92.9087 \pm 0.0782$ |
| ICA - PIX | $-3.4110 \pm 0.0024$ | $94.2135 \pm 0.0747$ |
| cHAD - PIX | $-3.5692 \pm 0.0045$ | $98.5839 \pm 0.0134$ |
| cSYM - PIX | $-3.5945 \pm 0.0047$ | $99.2815 \pm 0.0098$ |
| cICA - PIX | $-3.6205 \pm 0.0049$ | $100.0000 \pm 0.0000$ |

Table 1: Difference in ALL for gray value images with standard deviation over ten training and test set pairs. The column on the left displays the absolute difference to the PIX representation. The column on the right shows the relative difference with respect to the largest reduction achieved by ICA with non-factorial model.

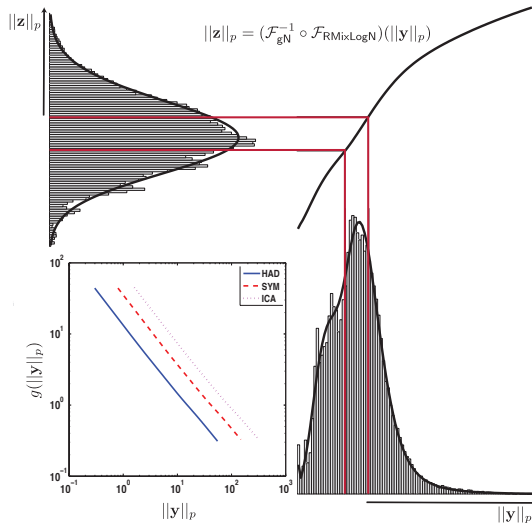

Figure 3: The curve in the upper right corner depicts the transformation $||\mathbf{z}||_p = (\mathcal{F}_{\mathsf{gN}}^{-1} \circ \mathcal{F}_{\mathsf{RMixLogN}})(||\mathbf{y}||_p)$ of the radial component in the ICA basis for gray scale images. The resulting radial distribution over $||\mathbf{z}||_p$ corresponds to the radial distribution of the $p$-generalized Normal. The inset shows the gain function $g(||\mathbf{y}||_p) = \frac{\mathcal{F}_{\mathsf{RMixLogN}}(||\mathbf{y}||_p)}{||\mathbf{y}||_p}$ in log-log coordinates. The scale parameter of the $p$-generalized normal was chosen such that the marginal had unit variance.

(HAD). By setting the other models in relation to the best joint model (cICA:= 100%), we are able to tell apart the relative contributions of bandpass filtering (HAD= 91%), particular filter shapes (SYM= 93%, ICA= 94%), contrast gain control (cHAD= 98.6%) as well as combined models (cSYM= 99%, cICA := 100%) to redundancy reduction (see Table 1). Thus, orientation selectivity (ICA) contributes less to the overall redundancy reduction than any model with contrast gain control (cHAD, cSYM, cICA). Additionally, the relative difference between the joint model (cICA) and plain contrast gain control (cHAD) is only about 1.4%. For cSYM it is even less, about 0.7%. The difference in redundancy reduction between center-surround filters and orientation selective filters becomes even smaller in combination with contrast gain control (1.3% for ICA vs. SYM, 0.7% for cICA vs. cSYM). However, it is still significant (t-test, $p = 5.5217 \cdot 10^{-9}$).

When examining the gain functions $g(||\mathbf{y}||_p) = \frac{(\mathcal{F}_{\mathsf{gN}}^{-1} \circ \mathcal{F}_{\mathsf{RMixLogN}})(||\mathbf{y}||_p)}{||\mathbf{y}||_p}$ resulting from the transformation of the radial components, we find that they approximately exhibit the form $g(||\mathbf{y}||_p) = \frac{c}{||\mathbf{y}||_p^\kappa}$. The inset in Figure 3 shows the gain control function $g(||\mathbf{y}||_p)$ in a log-log plot. While standard contrast gain control models assume $p = 2$ and $\kappa = 2$, we find that $\kappa$ between 0.90 and 0.93 to be optimal for redundancy reduction. $p$ depends on the shape of the linear filters and ranges from approximately 1.2 to 2. In addition, existing contrast gain models assume the form $g(||\mathbf{y}||_2) = \frac{1}{\sigma + ||y||_2^2}$, while we find that $\sigma$ must be approximately zero.

In the results above, the ICA filters always achieve the lowest ALL under both $p$-spherically symmetric models. For examining whether these filters really represent the best choice, we also optimized the filter shapes under the model of equation (2) via maximum likelihood estimation on the orthogonal group in whitened space [9; 18]. Figure 4 shows the filter shapes for ICA and the ones obtained from the optimization, where we used either the ICA solution or a random orthogonal matrix as starting point. Qualitatively, the filters look exactly the same. The ALL also changed just

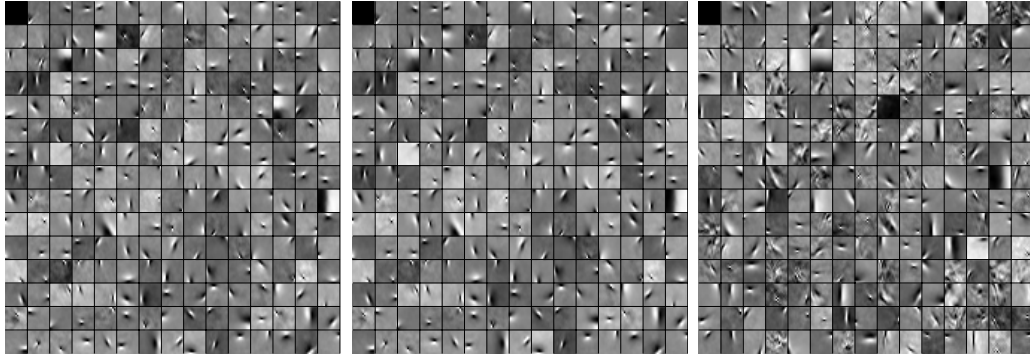

Figure 4: Filters optimized for ICA (*left*) and for the $p$-spherically symmetric model with radial mixture of Log-Normal distributions starting from the ICA solution (*middle*) and from a random basis (*right*). The first filter corresponds to the DC component, the others to the filter shapes under the respective model. Qualitatively the filter shapes are very similar. The ALL for the ICA basis under the mixture of Log-Normal model is $1.6748 \pm 0.0058$ bits/component (*left*), the ALL with the optimized filters is $1.6716 \pm 0.0056$ (*middle*) and $1.6841 \pm 0.0068$ (*right*).

marginally from $1.6748 \pm 0.0058$ to $1.6716 \pm 0.0056$ or $1.6841 \pm 0.0068$, respectively. Thus, the ICA filters are a stable and optimal solution under the model with contrast gain control, too.

## 4 Summary

In this report, we studied the conjoint effect of contrast gain control and orientation selectivity on redundancy reduction for natural images. In particular, we showed how the $L_p$-spherically distribution can be used to tune a nonlinearity of contrast gain control to remove higher-order redundancies in natural images.

The idea of using an $L_p$-spherically symmetric model for natural images has already been brought up by Hyvärinen and Köster in the context of Independent Subspace Analysis [15]. However, they do not use the $L_p$-distribution for contrast gain control, but apply a global contrast gain control filter on the images before fitting their model. They also use a less flexible $L_p$-distribution since their goal is to fit an ISA model to natural images and not to carry out a quantitative comparison as we did.

In our work, we find that the gain control function turns out to follow a power law, which parallels the classical model of contrast gain control. In addition, we find that edge filters also emerge in the non-linear model which includes contrast gain control. The relevance of orientation selectivity for redundancy reduction, however, is further reduced. In the linear framework (possibly endowed with a point-wise nonlinearity for each neuron) the contribution of orientation selectivity to redundancy reduction has been shown to be smaller than $5\%$ relative to whitening (i.e. bandpass filtering) alone [6; 10]. Here, we found that the contribution of orientation selectivity is even smaller than two percent relative to whitening plus gain control. Thus, this quantitative model comparison provides further evidence that orientation selectivity is not critical for redundancy reduction, while contrast gain control may play a more important role.

## Acknowledgements

The authors would like to thank Reshad Hosseini, Sebastian Gerwinn and Philipp Berens for fruitful discussions. This work is supported by the German Ministry of Education, Science, Research and Technology through the Bernstein award to MB (BMBF; FKZ: 01GQ0601), a scholarship of the German National Academic Foundation to FS, and the Max Planck Society.

## Footnotes

[1]Note that $||y||_p$ is only a norm in the strict sense if $p \geq 1$. However, since the following considerations also hold for $0 < p < 1$, we will employ the term "$p$-norm" and the notation "$||y||_p$" for notational convenience.

## References

[1] S. F. Arnold and J. Lynch. On Ali's characterization of the spherical normal distribution. *Journal of the Royal Statistical Society. Series B (Methodological)*, 44(1):49–51, 1982.

[2] J. J. Atick. Could information theory provide an ecological theory of sensory processing? *Network*, 3:213–251, 1992.

[3] F. Attneave. Informational aspects of visual perception. *Psychological Review*, 61:183–193, 1954.

[4] H. B. Barlow. Sensory mechanisms, the reduction of redundancy, and intelligence. In *The Mechanisation of Thought Processes*, pages 535–539, London: Her Majesty's Stationery Office, 1959.

[5] A. J. Bell and T. J. Sejnowski. The "independent components" of natural scenes are edge filters. *Vision Res.*, 37(23):3327–38, 1997.

[6] M. Bethge. Factorial coding of natural images: How effective are linear model in removing higher-order dependencies? *J. Opt. Soc. Am. A*, 23(6):1253–1268, June 2006.

[7] G. J. Brelstaff, A. Parraga, T. Troscianko, and D. Carr. Hyperspectral camera system: acquisition and analysis. In B. J. Lurie, J. J. Pearson, and E. Zilioli, editors, *Proceedings of SPIE*, volume 2587, pages 150–159, 1995. The database can be downloaded from: http://psy223.psy.bris.ac.uk/hyper/.

[8] G. Buchsbaum and A. Gottschalk. Trichromacy, opponent colours coding and optimum colour information transmission in the retina. *Proceedings of the Royal Society of London. Series B, Biological Sciences*, 220:89–113, November 1983.

[9] A. Edelman, T. A. Arias, and S. T. Smith. The geometry of algorithms with orthogonality constraints. *SIAM J. Matrix Anal. Appl.*, 20(2):303–353, 1999.

[10] J. Eichhorn, F. Sinz, and M. Bethge. Simple cell coding of natural images in V1: How much use is orientation selectivity? (arxiv:0810.2872v1). 2008.

[11] I. R. Goodman and S. Kotz. Mutltivariate $\theta$-generalized normal distributions. *Journal of Multivariate Analysis*, 3:204–219, 1973.

[12] A. K. Gupta and D. Song. $l_p$-norm spherical distribution. *Journal of Statistical Planning and Inference*, 60:241–260, 1997.

[13] D. J. Heeger. Normalization of cell responses in cat striate cortex. *Visual Neuroscience*, 9:181–198, 1992.

[14] A. Hyvärinen, J. Karhunen, and E. Oja. *Independent Component Analysis*. John Wiley & Sons, 2001.

[15] A. Hyvärinen and U. Köster. Complex cell pooling and the statistics of natural images. *Network*, 18:81–100, 2007.

[16] T.-W. Lee, T. Wachtler, and T. J. Sejnowski. Color opponency is an efficient representation of spectral properties in natural scenes. *Vision Res*, 42(17):2095–2103, Aug 2002.

[17] S. Lyu and E. P. Simoncelli. Nonlinear extraction of 'independent components' of elliptically symmetric densities using radial Gaussianization. Technical Report TR2008-911, Computer Science Technical Report, Courant Inst. of Mathematical Sciences, New York University, April 2008.

[18] J. H. Manton. Optimization algorithms exploiting unitary constraints. *IEEE Transactions on Signal Processing*, 50:635 – 650, 2002.

[19] B. A. Olshausen and D. J. Field. Emergence of simple-cell receptive field properties by learning a sparse code for natural images. *Nature*, 381:607–609, June 1996.

[20] O. Schwartz and E. P. Simoncelli. Natural signal statistics and sensory gain control. *Nature Neuroscience*, 4(8):819–825, August 2001.

[21] E. P. Simoncelli and O. Schwartz. Modeling surround suppression in V1 neurons with a statistically-derived normalization model. In M. S. Kearns, S. A. Solla, and D. A. Cohn, editors, *Adv. Neural Information Processing Systems (NIPS*98)*, volume 11, pages 153–159, Cambridge, MA, 1999. MIT Press.

[22] F. H. Sinz, S. Gerwinn, and M. Bethge. Characterization of the p-generalized normal distribution. *Journal of Multivariate Analysis*, 07/26/ 2008.

[23] D. Song and A. K. Gupta. $l_p$-norm uniform distribution. *Proceedings of the American Mathematical Society*, 125:595–601, 1997.

[24] J. H. van Hateren and A. van der Schaaf. Independent component filters of natural images compared with simple cells in primary visual cortex. *Proc R Soc Lond B Biol Sci.*, 265(1394):1724–1726, 1998.

[25] T Wachtler, T W Lee, and T J Sejnowski. Chromatic structure of natural scenes. *Journal of the Optical Society of America. A, Optics, image science, and vision*, 18:65–77, 2001. PMID: 11152005.

